# Variational Bounds for Mixed-Data Factor Analysis

**Mohammad Emtiyaz Khan**
University of British Columbia
Vancouver, BC, Canada V6T 1Z4
emtiyaz@cs.ubc.ca

**Guillaume Bouchard**
Xerox Research Center Europe
38240 Meylan, France
guillaume.bouchard@xerox.com

**Benjamin M. Marlin**
University of British Columbia
Vancouver, BC, Canada V6T 1Z4
bmarlin@cs.ubc.ca

**Kevin P. Murphy**
University of British Columbia
Vancouver, BC, Canada V6T 1Z4
murphyk@cs.ubc.ca

## Abstract

We propose a new variational EM algorithm for fitting factor analysis models with mixed continuous and categorical observations. The algorithm is based on a simple quadratic bound to the log-sum-exp function. In the special case of fully observed binary data, the bound we propose is significantly faster than previous variational methods. We show that EM is significantly more robust in the presence of missing data compared to treating the latent factors as parameters, which is the approach used by exponential family PCA and other related matrix-factorization methods. A further benefit of the variational approach is that it can easily be extended to the case of mixtures of factor analyzers, as we show. We present results on synthetic and real data sets demonstrating several desirable properties of our proposed method.

## 1 Introduction

Continuous latent factor models, such as factor analysis (FA) and probabilistic principal components analysis (PPCA), are very commonly used density models for continuous-valued data. They have many applications including latent factor discovery, dimensionality reduction, and missing data imputation. The factor analysis model asserts that a low-dimensional continuous latent factor $\mathbf{z}_n \in \mathbb{R}^L$ underlies each high-dimensional observed data vector $\mathbf{y}_n \in \mathbb{R}^D$. Standard factor analysis models assume the prior on the latent factor has the form $p(\mathbf{z}_n) = \mathcal{N}(\mathbf{z}_n | \mathbf{0}, \mathbf{I})$, while the likelihood has the form $p(\mathbf{y}_n | \mathbf{z}_n) = \mathcal{N}(\mathbf{y}_n | \mathbf{W}\mathbf{z}_n + \boldsymbol{\mu}, \boldsymbol{\Sigma})$. $\mathbf{W}$ is the $D \times L$ factor loading matrix, $\boldsymbol{\mu}$ is an offset term, and $\boldsymbol{\Sigma}$ is a $D \times D$ diagonal matrix specifying the marginal noise variances. If we set $\boldsymbol{\Sigma} = \sigma^2 \mathbf{I}$ and require $\mathbf{W}$ to be orthogonal, we recover probabilistic principal components analysis (PPCA). Such models can be easily fit using the expectation-maximization (EM) algorithm [Row97, TB99].

The FA model can be extended to other members of the exponential family by requiring that the natural (canonical) parameters have the form $\mathbf{W}\mathbf{z}_n + \boldsymbol{\mu}$ [WK01, CDS02, MHG08, LT10]. This is the unsupervised version of a generalized linear model (GLM), and is extremely useful since it allows for non-trivial dependencies between data variables with mixed types.

The principal difficulty with the general FA model is computational tractability, both at training and test time. A problem arises because the Gaussian prior on $p(\mathbf{z}_n)$ is not conjugate to the likelihood except when $\mathbf{y}_n$ also has a Gaussian distribution (the standard FA model). There are several approaches one can take to this problem. The simplest is to approximate the posterior $p(\mathbf{z}_n | \mathbf{y}_n)$ using a point estimate, which is equivalent to viewing the latent variables as parameters and estimating them by maximum likelihood. This approach is known as exponential family PCA (ePCA)

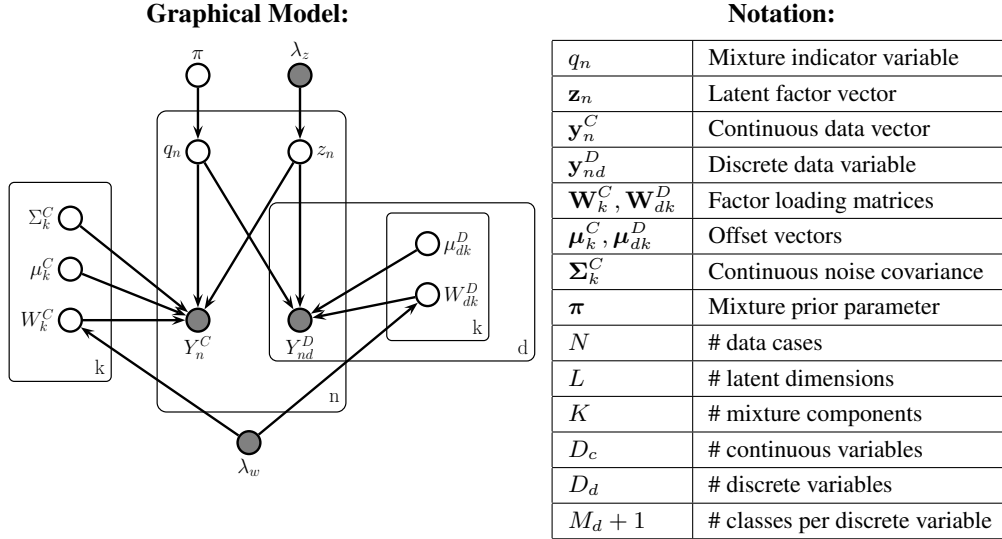

| **Graphical Model:** | | **Notation:** | |
|---|---|---|---|

| | |
|---|---|
| $q_n$ | Mixture indicator variable |
| $\mathbf{z}_n$ | Latent factor vector |
| $\mathbf{y}_n^C$ | Continuous data vector |
| $\mathbf{y}_{nd}^D$ | Discrete data variable |
| $\mathbf{W}_k^C, \mathbf{W}_{dk}^D$ | Factor loading matrices |
| $\boldsymbol{\mu}_k^C, \boldsymbol{\mu}_{dk}^D$ | Offset vectors |
| $\boldsymbol{\Sigma}_k^C$ | Continuous noise covariance |
| $\boldsymbol{\pi}$ | Mixture prior parameter |
| $N$ | # data cases |
| $L$ | # latent dimensions |
| $K$ | # mixture components |
| $D_c$ | # continuous variables |
| $D_d$ | # discrete variables |
| $M_d + 1$ | # classes per discrete variable |

*Figure 1:* The generalized mixture of factor analyzers model for discrete and continuous data.

[CDS02]. We refer to it as the "MM" approach to fitting the general FA model since we maximize over $\mathbf{z}_n$ in the E-step, as well as $\mathbf{W}$ in the M-step. The main drawback of the MM approach is that it ignores posterior uncertainty in $\mathbf{z}_n$, which can result in over-fitting unless the model is carefully regularized [WCS08]. This is a particular concern when we have missing data.

The opposite end of the model estimation spectrum is to integrate out both $\mathbf{z}_n$ and $\mathbf{W}$ using Markov chain Monte Carlo methods. This approach has recently been studied under the name "Bayesian exponential family PCA" [MHG08] using a Hamiltonian Monte Carlo (HMC) sampling approach. We will refer to this as the "SS" approach to indicate that we are integrating out both $\mathbf{z}_n$ and $\mathbf{W}$ by sampling. The SS approach preserves posterior uncertainty about $\mathbf{z}_n$ (unlike the MM approach) and is robust to missing data, but can have a significantly higher computational cost.

In this work, we study a variational EM model fitting approach that preserves posterior uncertainty about $\mathbf{z}_n$, is robust to missing data, and is more computationally efficient than SS. We refer to this as the "VM" approach to indicate that we integrate over $\mathbf{z}_n$ in the E-step after applying a variational bound, and maximize over $\mathbf{W}$ in the M-step. We focus on the case of continuous (Gaussian) and categorical data. Our main contribution is the development of variational EM algorithms for factor analysis and mixtures of factor analyzers based on a simple quadratic lower bound to the multinomial likelihood (which subsumes the Bernoulli case) [Boh92]. This bound results in an EM iteration that is computationally more efficient than the bound previously proposed by Jaakkola for binary PCA when the training data is fully observed [JJ96], but is less tight. The proposed bound has advantages relative to other previously introduced bounds, as we discuss in the following sections.

## 2 The Generalized Mixture of Factor Analyzers Model

In this section, we describe a model for mixed continuous and discrete data that we call the *generalized mixture of factor analyzers* model. This model has two important special cases: mixture models and factor analysis, both for mixed continuous and discrete data. We use the general model as well as both special cases in subsequent experiments. In this work, we focus on Gaussian distributed continuous data and multinomially distributed discrete data. The graphical model is given in Figure 1 while the probabilistic model is given in Equations 1 to 4. We begin with a description of the the general model and then highlight the two special cases.

We let $n \in \{1 \ldots N\}$ index data cases, $d \in \{1 \ldots D_d\}$ index discrete data dimensions and $k \in \{1 \ldots K\}$ index mixture components. Superscripts $C$ and $D$ indicate variables associated with continuous and discrete data respectively. We let $\mathbf{y}_n^C \in \mathbb{R}^{D_c}$ denote the continuous data vector and

$y_{nd}^D \in \{1 \ldots M+1\}$ denote the $d^{th}$ discrete data variable.[1] We use a 1-of-$(M+1)$ encoding for the discrete variables where a variable $y_{nd}^D = m$ is represented by a $(M+1)$-dimensional vector $\mathbf{y}_{nd}^D$ in which $m$'th element is set to 1, and all remaining elements equal 0. We denote the complete data vector by $\mathbf{y}_n = \left[\mathbf{y}_n^C, \mathbf{y}_{n1}^D, \ldots, \mathbf{y}_{nD_d}^D\right]$.

The generative process begins by sampling a state of the mixture indicator variable $q_n$ for each data case $n$ from a $K$-state multinomial distribution with parameters $\boldsymbol{\pi}$. Simultaneously, a length $L$ latent factor vector $\mathbf{z}_n \in \mathbb{R}^L$ is sampled from a zero-mean Gaussian distribution with precision parameter $\lambda_z$. Both steps are given in Equation 1. The natural parameters of the distribution over the data variables is obtained by passing the latent factor vector $\mathbf{z}_n$ through a linear function defined by a factor loading matrix and an offset term, both of which depend on the setting of the mixture indicator variable $q_n$.

$$p(\mathbf{z}_n, q_n | \boldsymbol{\theta}) = \mathcal{N}(\mathbf{z}_n | 0, \lambda_z^{-1} \mathbf{I}_L) \mathcal{M}(q_n | \boldsymbol{\pi}) \tag{1}$$

$$p(\mathbf{y}_n | \mathbf{z}_n, q_n = k, \boldsymbol{\theta}) = \mathcal{N}(\mathbf{y}_n^C | \mathbf{W}_k^C \mathbf{z}_n + \boldsymbol{\mu}_k^C, \boldsymbol{\Sigma}_k^C) \prod_{d=1}^{D_d} \mathcal{M}(\mathbf{y}_{nd}^D | \mathcal{S}(\boldsymbol{\eta}_{ndk})) \tag{2}$$

$$\boldsymbol{\eta}_{ndk} = \mathbf{W}_{dk}^D \mathbf{z}_n + \boldsymbol{\mu}_{dk}^D \tag{3}$$

$$\mathcal{S}_m(\boldsymbol{\eta}) = \exp[\eta_m - \mathrm{lse}(\boldsymbol{\eta})] \tag{4}$$

$$\mathrm{lse}(\boldsymbol{\eta}) = \log[\sum_{m=1}^{M+1} \exp(\eta_m)] \tag{5}$$

Assuming that $q_n = k$, the continuous data vector $\mathbf{y}_n^C$ is Gaussian distributed with mean $\mathbf{W}_k^C \mathbf{z}_n + \boldsymbol{\mu}_k^C$ and covariance $\boldsymbol{\Sigma}_k^C$, and each discrete data variable $\mathbf{y}_{nd}^D$ is multinomially distributed with natural parameters $\boldsymbol{\eta}_{ndk} = \mathbf{W}_{dk}^D \mathbf{z}_n + \boldsymbol{\mu}_{dk}^D$, as seen in Equation 2. Here, $\mathcal{N}(\cdot | \mathbf{m}, \mathbf{V})$ denotes a Gaussian distribution with mean $\mathbf{m}$ and covariance $\mathbf{V}$, while $\mathcal{M}(\cdot | \boldsymbol{\alpha})$ denotes a multinomial distribution with parameter vector $\boldsymbol{\alpha}$ such that $\sum_i \alpha_i = 1$ and $\alpha_i \geq 0$. For the discrete data variables, the natural parameter vector is converted into the standard mean parameter vector through the softmax function $\mathcal{S}(\boldsymbol{\eta}) = [\mathcal{S}_1(\boldsymbol{\eta}), \ldots, \mathcal{S}_{M+1}(\boldsymbol{\eta})]$, where $\mathcal{S}_m(\boldsymbol{\eta})$ is defined in Equation 4. The softmax function $\mathcal{S}_m(\boldsymbol{\eta})$ is itself defined in terms of the log-sum-exp (LSE) function, which we give in Equation 5.

We note that the factor loading matrices for the $k^{th}$ mixture component are $\mathbf{W}_k^C \in \mathbb{R}^{D_c \times L}$ and $\mathbf{W}_{dk}^D \in \mathbb{R}^{M+1 \times L}$, while the offsets are $\boldsymbol{\mu}_k^C \in \mathbb{R}^{D_c}$ and $\boldsymbol{\mu}_{dk}^D \in \mathbb{R}^{M+1}$. We define the ensemble of factor loading matrices and offsets to be $\mathbf{W}_k = [\mathbf{W}_k^C, \mathbf{W}_{1k}^D, \mathbf{W}_{2k}^D, \ldots, \mathbf{W}_{D_d k}^D]$ and $\boldsymbol{\mu}_k = [\boldsymbol{\mu}_k^C, \boldsymbol{\mu}_{1k}^D, \boldsymbol{\mu}_{2k}^D, \ldots, \boldsymbol{\mu}_{D_d k}^D]$, respectively. The complete set of parameters for this model is thus $\boldsymbol{\theta} = \{\mathbf{W}_{1:K}, \boldsymbol{\mu}_{1:K}, \boldsymbol{\Sigma}_{1:K}^C, \boldsymbol{\pi}, \lambda_z\}$. To complete the model specification, we must specify the prior on these parameters. For each row of each factor loading matrix $\mathbf{W}_k$, we use a Gaussian prior of the form $\mathcal{N}(\mathbf{0}, \lambda_w^{-1} \mathbf{I})$. We use vague conjugate priors for the remaining parameters.

As mentioned at the start of this section, this general model has two important special cases: generalized factor analysis and mixture models for mixed continuous and discrete data. The factor analysis model is obtained by using one mixture component and at least one latent factor ($K = 1, L > 1$). The mixture model is obtained by using no latent factors and at least one mixture component ($K > 1, L = 0$). In the mixture model case where $L = 0$, the distribution is modeled through the offset parameters $\boldsymbol{\mu}_k$ only. We will compare these three models in Section 5.

Before concluding this section, we point out one key difference between the current model and other latent factor models for discrete data like multinomial PCA [BJ04] and latent Dirichlet allocation (LDA) [BNJ03]. In our model, the natural parameters for discrete data are defined on a low-dimensional linear subspace and are mapped to the mean parameters via the softmax function. In multinomial PCA and LDA, the mean parameters are instead directly defined on a low-dimensional linear subspace. The latter approach can also be extended to the mixed-data case [BDdF+03]. However, model fitting is even more computationally challenging than in our approach. In fact, the bounds we propose can be used in this alternative setting, but we leave this to future work.

# 3 Variational Bounds for Model Fitting

In the standard expectation-maximization (EM) algorithm for mixtures of factor analyzers, the E-step consists of marginalizing over the complete-data log likelihood with respect to the posterior over the mixture indicator variable $q_n$ and latent factors $\mathbf{z}_n$. The M-step consists of maximizing the expected complete log likelihood with respect to the parameters $\boldsymbol{\theta}$. In the case of Gaussian observations, this posterior is available in closed form because of conjugacy. Introduction of discrete observations, however, makes it intractable to compute the posterior as the likelihood for these observations is not conjugate to the Gaussian prior on the latent factors.

To overcome these problems, we propose to use a quadratic bound on the LSE function. This allows us to obtain closed form updates for both the E and M steps. We use the quadratic bound described in [Boh92]. In rest of the paper, we will refer to it as the "Bohning bound". For simplicity, we describe the bound only for one discrete measurement with $K = 1$ and $\boldsymbol{\mu}_k = 0$ in order to suppress the $n$, $k$ and $d$ subscripts. To ensure identifiability, we assume that the last element of $\boldsymbol{\eta}$ is zero (this can be enforced by setting the last row of $\mathbf{W}$ to zero).

The key idea behind the Bohning bound is to take a second order Taylor series expansion of the LSE function around a point $\boldsymbol{\psi}$. An upper bound to the LSE function is found by replacing the Hessian matrix $\mathbf{H}(\boldsymbol{\psi})$, which appears in the second order term, with a fixed matrix $\mathbf{A}$ such that $\mathbf{A} - \mathbf{H}(\boldsymbol{\psi})$ is positive definite for all $\boldsymbol{\psi}$ [Boh92]. Bohning gives one such matrix $\mathbf{A}$, which we define below. The expansion point $\boldsymbol{\psi}$ is a free variational parameter that must be optimized.

$$\mathrm{lse}(\boldsymbol{\eta}) \leq \frac{1}{2}\boldsymbol{\eta}^T\mathbf{A}\boldsymbol{\eta} - \mathbf{b}_\psi^T\boldsymbol{\eta} + c_\psi \tag{6}$$

$$\mathbf{A} = \frac{1}{2}[\mathbf{I}_M - \mathbf{1}_M\mathbf{1}_M^T/(M+1)] \tag{7}$$

$$\mathbf{b}_\psi = \mathbf{A}\boldsymbol{\psi} - \mathcal{S}(\boldsymbol{\psi}) \tag{8}$$

$$c_\psi = \frac{1}{2}\boldsymbol{\psi}^T\mathbf{A}\boldsymbol{\psi} - \mathcal{S}(\boldsymbol{\psi})^T\boldsymbol{\psi} + \mathrm{lse}(\boldsymbol{\psi}) \tag{9}$$

$\boldsymbol{\psi} \in \mathbb{R}^M$ is the vector of variational parameters, $\mathbf{I}_M$ is the identity matrix of size $M \times M$ and $\mathbf{1}_M$ is a vector of ones of length $M$. By substituting this bound in to the log-likelihood, completing the square and exponentiating, we obtain the Gaussian lower bound described below. We obtain a Gaussian-like "pseudo" observation $\tilde{\mathbf{y}}_\psi$ corresponding to the discrete observation $\mathbf{y}^D$.

$$p(\mathbf{y}^D|\mathbf{z}, \mathbf{W}) \geq h(\boldsymbol{\psi})\mathcal{N}(\tilde{\mathbf{y}}_\psi|\mathbf{W}\mathbf{z}, \mathbf{A}^{-1}) \tag{10}$$

$$\tilde{\mathbf{y}}_\psi = \mathbf{A}^{-1}(\mathbf{b}_\psi + \mathbf{y}^D) \tag{11}$$

$$h(\boldsymbol{\psi}) = |2\pi\mathbf{A}^{-1}|^{\frac{1}{2}} \exp\left[\frac{1}{2}\tilde{\mathbf{y}}_\psi^T\mathbf{A}\tilde{\mathbf{y}}_\psi - c_\psi\right] \tag{12}$$

We use this result to obtain a lower bound for each mixed data vector $\mathbf{y}_n$. We will suppress the $\psi$ subscripts, which differ for each data point $n$ and each discrete variable $d$ for clarity. Let $\tilde{\mathbf{y}}_n = [\mathbf{y}_n^C, \tilde{\mathbf{y}}_{1,n}, \ldots, \tilde{\mathbf{y}}_{D_d,n}]$ be the data vector for a given $n$ and $\boldsymbol{\psi}$. It is straightforward to show that this observation gives the following lower bound on the joint likelihood,

$$p(\tilde{\mathbf{y}}_n|\mathbf{z}_n) = \mathcal{N}(\tilde{\mathbf{y}}_n|\tilde{\mathbf{W}}\mathbf{z}_n, \tilde{\boldsymbol{\Sigma}}), \quad \tilde{\mathbf{W}} = \left[\mathbf{W}^C, \mathbf{W}_1^D, \ldots, \mathbf{w}_{D_d}^D\right], \quad \tilde{\boldsymbol{\Sigma}} = \mathrm{diag}(\boldsymbol{\Sigma}^C, \mathbf{A}_1^{-1}, \ldots, \mathbf{A}_{D_d}^{-1})$$

Given this pseudo observation, the computation of the posterior means $\mathbf{m}_n$ and covariances $\mathbf{V}_n$ is similar to the Gaussian FA model as seen below. This result can be generalized to the mixture case in a straightforward way. The M-step is the same as in mixtures of Gaussian factor analyzers [GH96].

$$\mathbf{V}_n = (\tilde{\mathbf{W}}^T\tilde{\boldsymbol{\Sigma}}^{-1}\tilde{\mathbf{W}} + \lambda_z\mathbf{I}_L)^{-1}, \quad \mathbf{m}_n = \mathbf{V}_n\tilde{\mathbf{W}}^T\tilde{\boldsymbol{\Sigma}}^{-1}\tilde{\mathbf{y}}_n \tag{13}$$

The only question remaining is how to obtain the value of $\boldsymbol{\psi}$. By maximizing the lower bound, one can show that the optimal value is $\boldsymbol{\psi}_n = \tilde{\mathbf{W}}\mathbf{m}_n$. This follows from the fact that the Bohning bound is tight for $\mathrm{lse}(\boldsymbol{\eta})$ when $\boldsymbol{\psi} = \boldsymbol{\eta}$, and that the curvature is independent of $\boldsymbol{\eta}$ [Boh92]. We iterate this update until convergence. In practice, we find that the method usually converges in five or fewer iterations.

The most attractive feature of the bound described above is its computational efficiency. To see this, note that the posterior covariance $\mathbf{V}_n$ does not in fact depend on $n$ if the data vector is fully

observed, since $\mathbf{A}$ is a constant matrix. Consequently we need only invert $\mathbf{V}_n$ once outside the EM loop instead of $N$ times, once for each data point. We will see in the next section that the other existing quadratic bounds do not have this property. To derive the overall computational cost of our EM algorithm, let us define the total dimension of $\tilde{\mathbf{y}}_n$ to be $D$ and assume $K = 1$. Computing $\mathbf{V}_n$ takes $O(L^3 + L^2 D)$ time, and computing each $\mathbf{m}_n$ takes $O(L^2 + LD)$ time. So the total cost of one E-step is $O(L^3 + L^2 D + NI(L^2 + LD))$, where $I$ is the number of variational updates. If there is missing data, $\mathbf{V}_n$ will change across data cases, so the total cost will be $O(NI(L^3 + L^2 D))$.

## 3.1 Comparison with Other Bounding Methods

In the binary case, the Bohning bound reduces to the following: $\log(1 + e^\eta) \leq \frac{1}{2} A \eta^2 - b_\psi \eta + c_\psi$, where $A = 1/4$, $b_\psi = A\psi - (1 + e^{-\psi})^{-1}$, and $c_\psi = \frac{1}{2} A \psi^2 - (1 + e^{-\psi})^{-1} \psi + \log(1 + e^\psi)$. It is interesting to compare this bound to Jaakkola's bound [JJ96] used in [Tip98, YT04]. This bound can also be written in the quadratic form: $\log(1 + e^\eta) \leq \frac{1}{2} \tilde{A}_\xi \eta^2 - \tilde{b}_\xi \eta + \tilde{c}_\xi$, where $\tilde{A}_\xi = 2\lambda_\xi$, $\tilde{b}_\xi = -\frac{1}{2}$, $\tilde{c}_\xi = -\lambda_\xi \xi^2 - \frac{1}{2}\xi + \log(1 + e^\xi)$, $\lambda_\xi = \frac{1}{2\xi}(\frac{1}{1 + e^{-\xi}} - \frac{1}{2})$.

Although the Jaakkola bound is tighter than the Bohning bound, it has higher computational complexity. The reason is that the $\tilde{A}_\xi$ parameter depends on $\xi$ and hence on $n$, which means we need to compute a different posterior covariance matrix for each $n$. Consequently, the cost of an E-step is $O(NI(L^3 + L^2 D))$, even if there is no missing data (note the $L^3$ term inside the $NI$ loop).

To explore the speed vs accuracy trade-off, we use the synthetic binary data described in [MHG08] with $N = 600$, $D = 16$, and $10\%$ missing data. We learn a binary FA model with $L = 10$, $\lambda_z = 1$, and $\lambda_w = 0$. We learn on the observed entries in the data matrix and compute the mean squared error (MSE) on the held out missing entries as in [MHG08]. We average the results over 20 repetitions of the experiment. We see in Figure 2 (top left) that the Jaakkola bound gives a lower MSE than Bohning's bound in less time on this data. Next, we consider the case where the training data is fully observed using a modified version of the data generating procedure described in [MHG08]. We vary $D$ from 16 to 128 while setting $L = 0.25D$ and $N = 10D$. We sample $L$ different binary prototypes at random, assign each data case to a prototype, and add $10\%$ random binary noise. We measure the average time per iteration over 40 iterations of each method. Figure 2 (bottom left) shows that the Bohning bound exhibits much better scalability per iteration than the Jaakkola bound in this regime.

The speed issue becomes more serious when combining binary variables with categorical variables. Firstly, there is no direct extension of the Jaakkola bound to the general categorical case. Hence, to combine categorical variables with binary variables, we can use the Jaakkola bound for binary and the Bohning for the rest. However, this is not computationally efficient as we need to compute the posterior covariance for each data point because of the Jaakkola bound. For computational simplicity, we use Bohning's bound for both binary and categorical data.

Various other bounds and approximations to the multinomial likelihood also exist; however, they are all more computationally intensive, and do not give an efficient variational algorithm. To the best of our knowledge these methods have not been applied to the FA model, but we describe them briefly for completeness. An extension of the Jaakkola bound to the multinomial case was given in [Bou07]. However, this tends to be less accurate than the Bohning bound. Another approach [BL06] is to use the concavity of the log function to write $\mathrm{lse}(\boldsymbol{\eta}) \leq \nu(1 + \sum_{j=1}^{M} \exp(\eta_j)) - \log \nu - 1$, where $\nu$ is a variational parameter. This bound does not give closed form updates for the E and M steps so a numerical optimizer needs to be used (see [BL06] for details).

Instead of using a bound, an alternative approach is to apply a quadratic approximation derived from a Taylor series expansion of the LSE function [AX07]. This provides a tighter approximation that could perform better than a bound, but one cannot make convergence guarantees when using it inside of EM. In practice we found this alternative approach to be very slow on the datasets that we consider. In view of its speed and simplicity, we will only consider the Bohning method for the remainder of the paper.

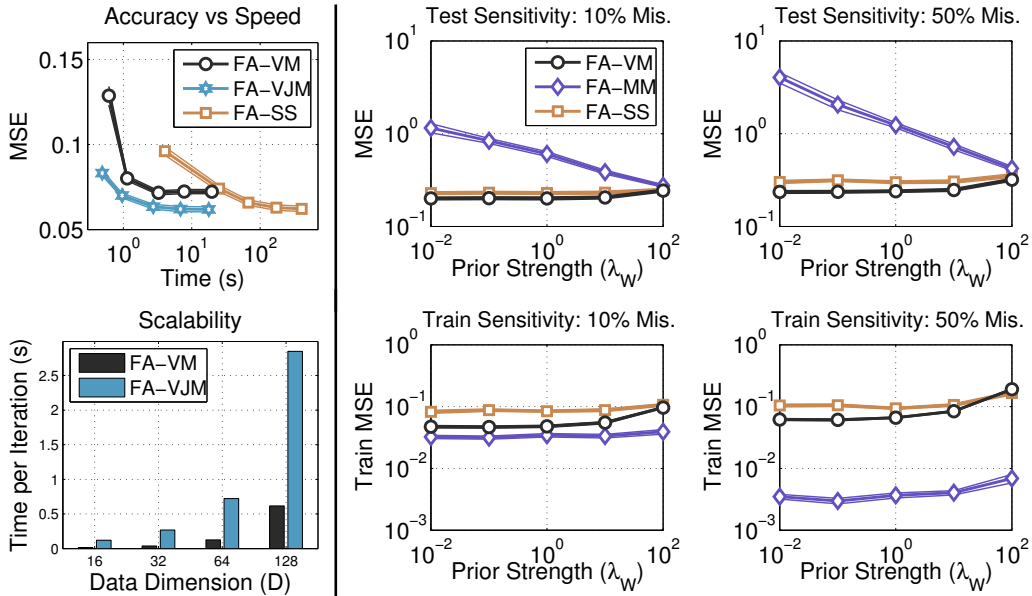

*Figure 2:* Top left: accuracy vs speed of variational EM with the Bohning bound (FA-VM), Jaakkola bound (FA-VJM) and HMC (FA-SS) on synthetic binary data. Bottom left: Time per iteration of EM with Bohning bound and Jaakkola bound as we vary $D$. Right: MSE vs $\lambda_w$ for FA-MM, FA-VM, and FA-SS on synthetic Gaussian data. We show results on the test and training sets, for 10% and 50% missing data.

## 4 Alternative Estimation Approaches

In this section, we discuss several alternative methods for fitting the generalized FA model in the case $K = 1$, which we compare to the VM method. We defer comparisons of FA to mixture models to Section 5.

### 4.1 Maximize-Maximize (MM) Method

The simplest approach to fit the FA model is to maximize $\log p(\mathbf{Y}, \mathbf{Z}, \mathbf{W}|\lambda_w, \lambda_z)$ with respect to $\mathbf{Z}$ and $\mathbf{W}$, the matrix of latent factor values and the factor loading matrix. It is straightforward to compute the gradient of the log posterior and apply a generic optimizer (we use the limited-memory quasi-newton method). Alternatively, one can use coordinate descent [CDS02]. We set the hyper-parameters $\lambda_w$ and $\lambda_z$ by cross validation. To handle missing data, we simply evaluate the gradients by only summing over the observed entries of $\mathbf{Y}$. At test time, consider a data vector consisting of missing and observed components, $\mathbf{y}_* = [\mathbf{y}_{*m}, \mathbf{y}_{*o}]$. To fill in the missing entries, we compute $\hat{\mathbf{z}}_* = \arg\max p(\mathbf{z}_*, \mathbf{y}_{*o}|\hat{\mathbf{W}})$ and use it with $\hat{\boldsymbol{\theta}}$ to predict $\mathbf{y}_{*m}$.

The MM approach is simple and widely applicable, but these benefits come at the expense of ignoring the posterior variance of $\mathbf{Z}$ [WCS08]. This has negative consequences for the method in terms of sensitivity to the parameters $\lambda_w$ and $\lambda_z$. To illustrate this effect, we generate a continuous dataset using $D = 10$, $L = 5$, and $N = 200$ data cases by sampling from the FA model. We set $\lambda_w = 1$, $\lambda_z = 1$, and $\sigma_c = 0.1$. We standardize each data dimension to have unit variance and zero mean. We consider the case of $10\%$ and $50\%$ missing data. We evaluate the sensitivity of the methods to the setting of the posterior precision parameter $\lambda_w$ by varying it over the range $10^{-2}$ to $10^2$. We fix $\lambda_z = 1$, since this is the standard assumption when fitting FA models. We run the methods on a random $50/50$ train/test split. We train on the observed entries in the training set, and then compute MSE on the missing entries in the training and test sets. We average the results over 20 repetitions of the experiment.

Figure 2 (top right) shows that the test MSE of the MM method is extremely sensitive to the prior precision $\lambda_w$. We can see that this sensitivity increases as a function of the missing data rate. We hypothesize that this is a result of the MM method ignoring the posterior uncertainty in $\mathbf{Z}$.

This is supported by looking at the MSE on the training set, Figure 2 (bottom right). We see that the MM method overfits when $\lambda_w$ is small. Consequently, MM requires a careful discrete search over the values of $\lambda_w$, which is slow, since the quality of each such value must be estimated by cross-validation. By contrast, the VM method takes the posterior uncertainty about $\mathbf{Z}$ into account, resulting in almost no sensitivity to $\lambda_w$ over this range. Henceforth we set $\lambda_w = 0$ for VM, meaning we are performing (approximate) maximum likelihood parameter estimation.

## 4.2 Sample-Sample (SS) Method

An alternative to the MM approach is to sample both $\mathbf{Z}$ and $\mathbf{W}$ from their posteriors using Hamiltonian Monte Carlo (HMC) [MHG08]. We call this the "SS" method, since we sample both $\mathbf{Z}$ and $\mathbf{W}$. HMC leverages the fact that we can compute the gradient of the log posterior in closed form. However, it has several important parameters that must be set including the step size, the momentum distribution, the number of leapfrog steps, etc.

To handle missing data, we can simply evaluate the gradients by only summing over the observed entries of $\mathbf{Y}$. We do not need to impute the missing entries on the training set. At test time, we have a collection of samples of $\mathbf{W}$. For each sample of $\mathbf{W}$ and each test case, we sample a set of $\mathbf{z}$, and compute an averaged prediction for $\mathbf{y}_m$. In Figure 2 (right), we see that SS is insensitive to $\lambda_w$, just like VM, since it also models posterior uncertainty in $\mathbf{Z}$ (note that the absolute MSE values are higher for SS than VM since for continuous data, VM corresponds to EM with an exact posterior). However, in Figure 2 (top left), we see that SS can be much slower than VM. In the remainder of the paper we focus on deterministic fitting methods only.

## 5 Experiments on Real Data

In this section, we evaluate the performance of our model on real data with mixed continuous and discrete variables. We consider the following three cases of our model: (1) a model with latent factors but no mixtures (FA) (2) a model with mixtures but no latent factors (Mix) and (3) the general mixture of factor analyzers model (MixFA). To learn the FA model, we consider the FA-MM and FA-VM approaches. For the Mix model, we use the standard EM algorithm. In the Mix model, continuous variables can be modeled with either a diagonal or a full covariance matrix. We refer to these two variants as Mix-Diag and Mix-Full. For MixFA model, we use the VM approach. This gives us five methods: FA-MM, FA-VM, MixFA, Mix-Full and Mix-Diag.

We consider three real datasets of different sizes (see the table in Figure 3).[2] For each dataset, we use 70% for training, 10% for validation and 20% for testing. We consider 20 splits for each dataset. We use the validation set to determine the number of latent factors and the number of mixtures (ranges shown in the table) with imputation error (described below) as our performance objective. For the FA-MM method, we set the values of the regularization parameters $\lambda_z$ and $\lambda_w$ by cross validation. We use the range $\{0.01, 0.1, 1, 10, 100\}$ for both $\lambda_z$ and $\lambda_w$. As VM is robust to the setting of these parameters, we set $\lambda_z = 1$ and $\lambda_w = 0$.

One way to assess the performance of a generative model is to see how well it can impute missing data. We do this by randomly introducing missing values in the test data with a missing data rate of 0.3. For continuous variables, we compute the imputation MSE averaged over all the missing values (these variables are standardized beforehand). For discrete variables, we report the cross-entropy (averaged over missing values) defined as $\mathbf{y}^T \log \hat{\mathbf{p}}$, where $\hat{p}_m$ is the estimated probability that $\mathbf{y} = m$ and $\mathbf{y}$ uses the one-of-$(M+1)$ encoding.

These errors are shown in Figure 3 along with the running time for ASES dataset in the bottom right subfigure. We see that FA-VM consistently performs better than FA-MM for all the datasets. Moreover, because of the need for cross-validation, FA-MM takes more time than FA-VM. We also see that the Mix model, although faster, performs worse than FA-VM. Finally, as expected, MixFA generally performs slightly better than FA, but takes longer to run.

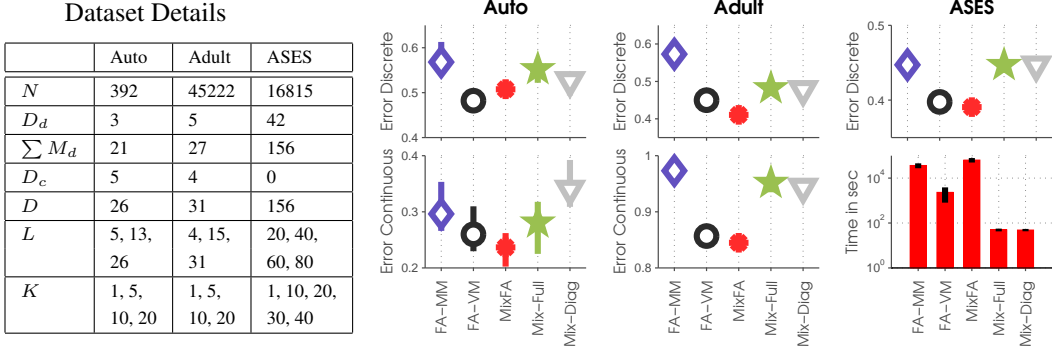

| Dataset Details | | | |
|---|---|---|---|
| | Auto | Adult | ASES |
| $N$ | 392 | 45222 | 16815 |
| $D_d$ | 3 | 5 | 42 |
| $\sum M_d$ | 21 | 27 | 156 |
| $D_c$ | 5 | 4 | 0 |
| $D$ | 26 | 31 | 156 |
| $L$ | 5, 13, 26 | 4, 15, 31 | 20, 40, 60, 80 |
| $K$ | 1, 5, 10, 20 | 1, 5, 10, 20 | 1, 10, 20, 30, 40 |

*Figure 3:* Left: the table shows the details of each dataset used. Here $D = D_c + \sum M_d$ is the total size of the data vector. $L$ and $K$ are the ranges of number of latent factors and mixture components used for cross validation. Note that the maximum value of $L$ is $D$, as required by the FA model. Right: the figure shows the imputation error for each dataset for continuous and discrete variables. The bottom right subfigure shows the timing comparison for the ASES dataset.

# 6 Discussion and Future Work

In this work we have proposed a new variational EM algorithm for fitting factor analysis models with mixed data. The algorithm is based on the Bohning bound, a simple quadratic bound to the log-sum-exp function. In the special case of fully observed binary data, the Bohning bound iteration is theoretically faster than Jaakkola's bound iteration and we have demonstrated this advantage empirically. More importantly, the Bohning bound also easily extends to the categorical case. This enables, for the first time, an efficient variational method for fitting a factor analysis model to mixed continuous, binary, and categorical observations.

In comparison to the maximize-maximize (MM) method, which forms the basis of ePCA and other matrix factorization methods, our variational EM method accounts for posterior uncertainty in the latent factors, leading to reduced sensitivity to hyper parameters. This has important practical consequences as the MM method requires extensive cross validation while our approach does not.

We have compared a range of models and algorithms in terms of imputation performance on real data. This analysis shows that the cost of the cross validation search for MM is higher than the cost of fitting the FA model using our method. It also shows that standard alternatives to FA, such as finite mixture models, do not perform as well as FA. Finally, we show that the MixFA model can yield a performance improvement over a single FA model, although at a higher computational cost.

We note that the quadratic bound that we study can be used in a variety of other models, such as linear-Gaussian state-space models with categorical observations [SH03]. It might be an interesting alternative to a Laplace approximation to the posterior, which is used in [KPBSK10, RMC09]. The bound might also be useful in the context of the correlated topic model [BL06, AX07], where similar variational EM methods have been applied.

In the Bayesian statistics literature, it is common to use latent factor models combined with a probit observation model; this allows one to perform inference for the latent states using efficient auxiliary-variable MCMC techniques (see e.g., [HSC09, Dun07]). Additionally, the recently proposed Riemannian Manifold Hamiltonian Monte Carlo sampler [GCC09] may significantly speed-up sampling-based approaches for mixed-data factor analysis models. We leave a comparison to these approaches to future work.

### Acknowledgments

We would like to thank the reviewers for their helpful coments. This work was completed in part at the Xerox Research Center Europe and was supported by the Pacific Institute for the Mathematical Sciences and the Killam Trusts at the University of British Columbia.

## Footnotes

[1] Note that we assume all the discrete data variables have the same number of states, namely $M+1$, for notational simplicity only. In the general case, the $d^{th}$ discrete variable has $M_d + 1$ states.

[2]Adult and Auto are available in UCI repository, while ASES dataset is a subset of Asia-Europe Survey from www.icpsr.umich.edu

# References

[AX07] A. Ahmed and E. Xing. On tight approximate inference of the logistic-normal topic admixture model. In *AI/Statistics*, 2007.

[BDdF+03] Kobus Barnard, Pinar Duygulu, Nando de Freitas, David Forsyth, David Blei, and Michael I. Jordan. Matching words and pictures. *J. of Machine Learning Research*, 3:1107–1135, 2003.

[BJ04] W. Buntine and A. Jakulin. Applying Discrete PCA in Data Analysis. In *UAI*, 2004.

[BL06] D. Blei and J. Lafferty. Correlated topic models. In *NIPS*, 2006.

[BNJ03] D. Blei, A. Ng, and M. Jordan. Latent dirichlet allocation. *J. of Machine Learning Research*, 3:993–1022, 2003.

[Boh92] D. Bohning. Multinomial logistic regression algorithm. *Annals of the Inst. of Statistical Math.*, 44:197–200, 1992.

[Bou07] G. Bouchard. Efficient bounds for the softmax and applications to approximate inference in hybrid models. In *NIPS 2007 Workshop on Approximate Inference in Hybrid Models*, 2007.

[CDS02] M. Collins, S. Dasgupta, and R. E. Schapire. A generalization of principal components analysis to the exponential family. In *NIPS-14*, 2002.

[Dun07] D. Dunson. Bayesian methods for latent trait modelling of longitudinal data. *Stat. Methods Med. Res.*, 16(5):399–415, Oct 2007.

[GCC09] M. Girolami, B. Calderhead, and S.A. Chin. Riemannian manifold hamiltonian monte carlo. *Arxiv preprint arXiv:0907.1100*, 2009.

[GH96] Z. Ghahramani and G. Hinton. The EM algorithm for mixtures of factor analyzers. Technical report, Dept. of Comp. Sci., Uni. Toronto, 1996.

[HSC09] P. R. Hahn, J. Scott, and C. Carvahlo. Sparse Factor-Analytic Probit Models. Technical report, Duke, 2009.

[JJ96] T. Jaakkola and M. Jordan. A variational approach to Bayesian logistic regression problems and their extensions. In *AI/Statistics*, 1996.

[KPBSK10] S. Koyama, L. Perez-Bolde, C. Shalizi, and R. Kass. Approximate methods for state-space models. Technical report, CMU, 2010.

[LT10] J. Li and D. Tao. Simple exponential family PCA. In *AI/Statistics*, 2010.

[MHG08] S. Mohamed, K. Heller, and Z. Ghahramani. Bayesian Exponential Family PCA. In *NIPS*, 2008.

[RMC09] H. Rue, S. Martino, and N. Chopin. Approximate Bayesian Inference for Latent Gaussian Models Using Integrated Nested Laplace Approximations. *J. of Royal Stat. Soc. Series B*, 71:319–392, 2009.

[Row97] S. Roweis. EM algorithms for PCA and SPCA. In *NIPS*, 1997.

[SH03] V. Siivola and A. Honkela. A state-space method for language modeling. In *Proc. IEEE Workshop on Automatic Speech Recognition and Understanding (ASRU)*, pages 548–553, 2003.

[TB99] M. Tipping and C. Bishop. Probabilistic principal component analysis. *J. of Royal Stat. Soc. Series B*, 21(3):611–622, 1999.

[Tip98] M. Tipping. Probabilistic visualization of high-dimensional binary data. In *NIPS*, 1998.

[WCS08] Max Welling, Chaitanya Chemudugunta, and Nathan Sutter. Deterministic latent variable models and their pitfalls. In *Intl. Conf. on Data Mining*, 2008.

[WK01] Michel Wedel and Wagner Kamakura. Factor analysis with (mixed) observed and latent variables in the exponential family. *Psychometrika*, 66(4):515–530, December 2001.

[YT04] K. Yu and V. Tresp. Heterogenous data fusion via a probabilistic latent-variable model. In *Organic and Pervasive Computing (ARCS 2004)*, 2004.

